# SPEECH RECOGNITION EXPERIMENTS
# WITH PERCEPTRONS

D. J. Burr
Bell Communications Research
Morristown, NJ 07960

## ABSTRACT

Artificial neural networks (ANNs) are capable of accurate recognition of simple speech vocabularies such as isolated digits [1]. This paper looks at two more difficult vocabularies, the alphabetic E-set and a set of polysyllabic words. The E-set is difficult because it contains weak discriminants and polysyllables are difficult because of timing variation. Polysyllabic word recognition is aided by a time pre-alignment technique based on dynamic programming and E-set recognition is improved by focusing attention. Recognition accuracies are better than 98% for both vocabularies when implemented with a single layer perceptron.

## INTRODUCTION

Artificial neural networks perform well on simple pattern recognition tasks. On speaker trained spoken digits a layered network performs as accurately as a conventional nearest neighbor classifier trained on the same tokens [1]. Spoken digits are easy to recognize since they are for the most part monosyllabic and are distinguished by strong vowels.

It is reasonable to ask whether artificial neural networks can also solve more difficult speech recognition problems. Polysyllabic recognition is difficult because multi-syllable words exhibit large timing variation. Another difficult vocabulary, the alphabetic E-set, consists of the words B, C, D, E, G, P, T, V, and Z. This vocabulary is hard since the distinguishing sounds are short in duration and low in energy.

We show that a simple one-layer perceptron [7] can solve both problems very well if a good input representation is used and sufficient examples are given. We examine two spectral representations — a smoothed FFT (fast Fourier transform) and an LPC (linear prediction coefficient) spectrum. A time stabilization technique is described which pre-aligns speech templates based on peaks in the energy contour. Finally, by focusing attention of the artificial neural network to the beginning of the word, recognition accuracy of the E-set can be consistently increased.

A layered neural network, a relative of the earlier perceptron [7], can be trained by a simple gradient descent process [8]. Layered networks have been

applied successfully to speech recognition [1], handwriting recognition [2], and to speech synthesis [11]. A variation of a layered network [3] uses feedback to model causal constraints, which can be useful in learning speech and language. Hidden neurons within a layered network are the building blocks that are used to form solutions to specific problems. The number of hidden units required is related to the problem [1,2]. Though a single hidden layer can form any mapping [12], no more than two layers are needed for disjunctive normal form [4]. The second layer may be useful in providing more stable learning and representation in the presence of noise. Though neural nets have been shown to perform as well as conventional techniques[1,5], neural nets may do better when classes have outliers [5].

<center>PERCEPTRONS</center>

A simple perceptron contains one input layer and one output layer of neurons directly connected to each other (no hidden neurons). This is often called a one-layer system, referring to the single layer of weights connecting input to output. Figure 1. shows a one-layer perceptron configured to sense speech patterns on a two-dimensional grid. The input consists of a 64-point spectrum at each of twenty time slices. Each of the 1280 inputs is connected to each of the output neurons, though only a sampling of connections are shown. There is one output neuron corresponding to each pattern class. Neurons have standard linear-weighted inputs with logistic activation.

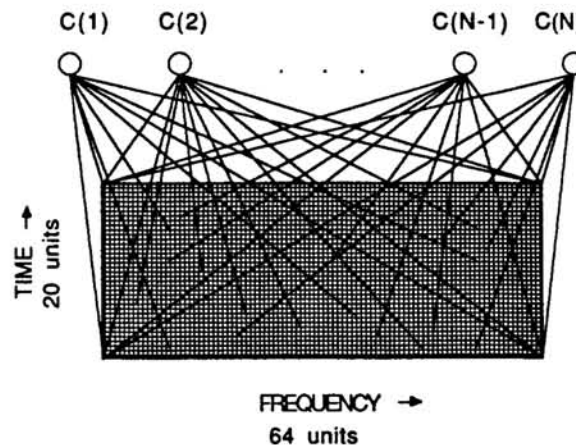

Figure 1. A single layer perceptron sensing a time-frequency array of sample data. Each output neuron $C(i)$ $(1 \leq i \leq N)$ corresponds to a pattern class and is full connected to the input array (for clarity only a few connections are shown).

An input word is fit to the grid region by applying an automatic endpoint detection algorithm. The algorithm is a variation of one proposed by Rabiner and Sambur [9] which employs a double threshold successive approximation

method. Endpoints are determined by first detecting threshold crossings of energy and then of zero crossing rate. In practice a level crossing other than zero is used to prevent endpoints from being triggered by background sounds.

## INPUT REPRESENTATIONS

Two different input representations were used in this study. The first is a Fourier representation smoothed in both time and frequency. Speech is sampled at 10 KHz and Hamming windowed at a number of sample points. A 128-point FFT spectrum is computed to produce a template of 64 spectral samples at each of twenty time frames. The template is smoothed twice with a time window of length three and a frequency window of length eight.

For comparison purposes an LPC spectrum is computed using a tenth order model on 300-sample Hamming windows. Analysis is performed using the autocorrelation method with Durbin recursion [6]. The resulting spectrum is smoothed over three time frames.

Sample spectra for the utterance "neural-nets" is shown in Figure 2. Notice the relative smoothness of the LPC spectrum which directly models spectral peaks.

FFT                                                  LPC

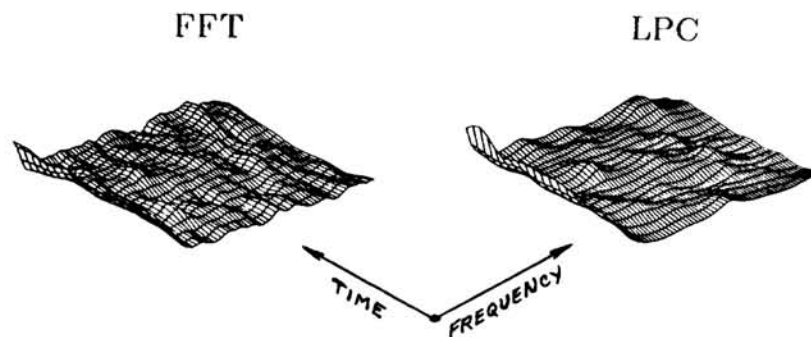

Figure 2. FFT and LPC time-frequency plots for the utterance "neural nets". Time is toward the left, and frequency, toward the right.

## DYNAMIC TIME ALIGNMENT

Conventional speech recognition systems often employ a time normalization technique based on dynamic programming [10]. It is used to warp the time scales of two utterances to obtain optimal alignment between their spectral frames. We employ a variation of dynamic programming which aligns energy contours rather than spectra. A reference energy template is chosen for each pattern class, and incoming patterns are warped onto it. Figure 3 shows five utterances of "neural-nets" both before and after time alignment. Notice the improved alignment of energy peaks.

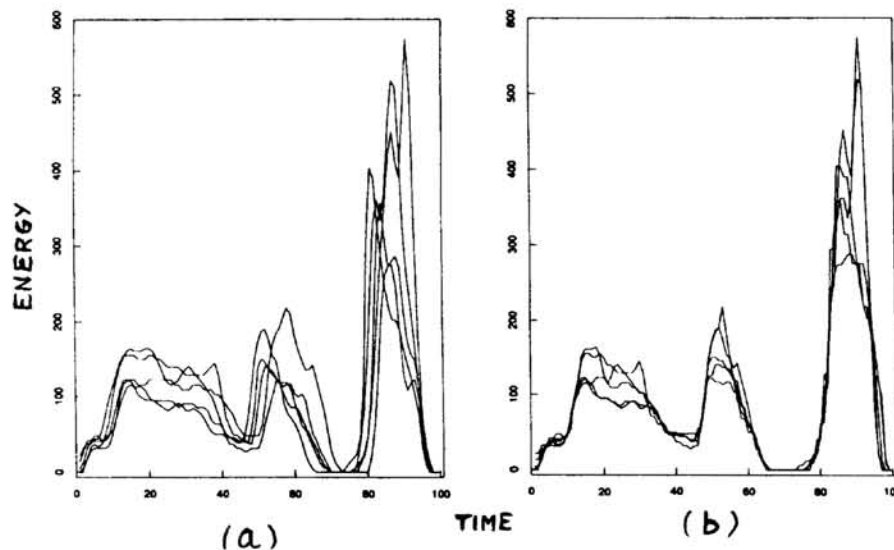

Figure 3. (a) Superimposed energy plots of five different utterances of "neural nets". (b). Same utterances after dynamic time alignment.

## POLYSYLLABLE RECOGNITION

Twenty polysyllabic words containing three to five syllables were chosen, and five tokens of each were recorded by a single male speaker. A variable number of tokens were used to train a simple perceptron to study the effect of training set size on performance. Two performance measures were used: classification accuracy, and an RMS error measure. Training tokens were permuted to obtain additional experimental data points.

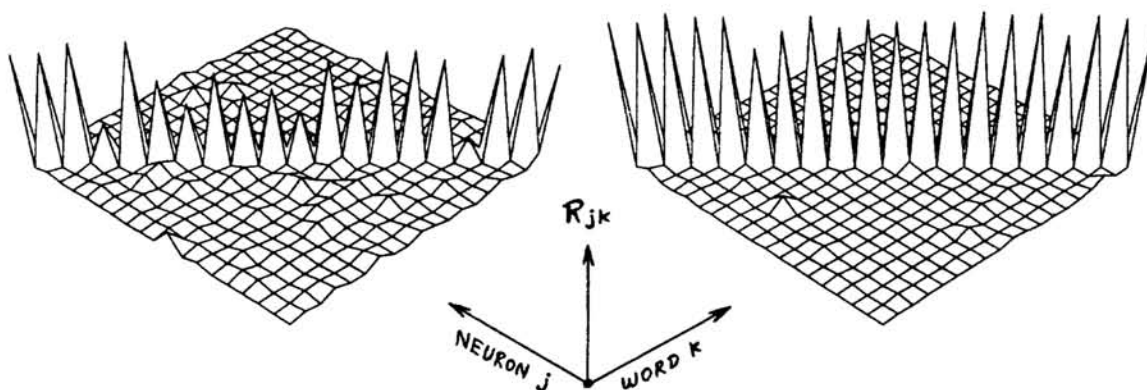

Figure 4. Output responses of a perceptron trained with one token per class (left) and four tokens per class (right).

Figure 4 shows two representative perspective plots of the output of a perceptron trained on one and four tokens respectively per class. Plots show network response (z-coordinate) as a function of output node (left axis) and test word index (right axis). Note that more training tokens produce a more ideal map — a map should have ones along the diagonal and zeroes everywhere else.

Table 1 shows the results of these experiments for three different representations: (1) FFT, (2) LPC and (3) time aligned LPC. This table lists classification accuracy as a function of number of training tokens and input representation. The perceptron learned to classify the unseen patterns perfectly for all cases except the FFT with a single training pattern.

| Table 1. Polysyllabic Word Recognition Accuracy | | | | |
|---|---|---|---|---|
| Number Training Tokens | 1 | 2 | 3 | 4 |
| FFT | 98.7% | 100% | 100% | 100% |
| LPC | 100% | 100% | 100% | 100% |
| Time Aligned LPC | 100% | 100% | 100% | 100% |
| Permuted Trials | 400 | 300 | 200 | 100 |

A different performance measure, the RMS error, evaluates the degree to which the trained network output responses $R_{jk}$ approximate the ideal targets $T_{jk}$. The measure is evaluated over the $N$ non-trained tokens and $M$ output nodes of the network. $T_{jk}$ equals 1 for $j=k$ and 0 for $j \neq k$.

$$RMS\ Error = \frac{\left[ \sum_{j=1}^{M} \sum_{k=1}^{N} (T_{jk} - R_{jk})^2 \right]^{1/2}}{MN}$$

Figure 5 shows plots of RMS error as a function of input representation and training patterns. Note that the FFT representation produced the highest error, LPC was about 40% less, and time-aligned LPC only marginally better than non-aligned LPC. In a situation where many choices must be made (i.e. vocabularies much larger than 20 words) LPC is the preferred choice, and time alignment could be useful to disambiguate similar words. Increased number of training tokens results in improved performance in all cases.

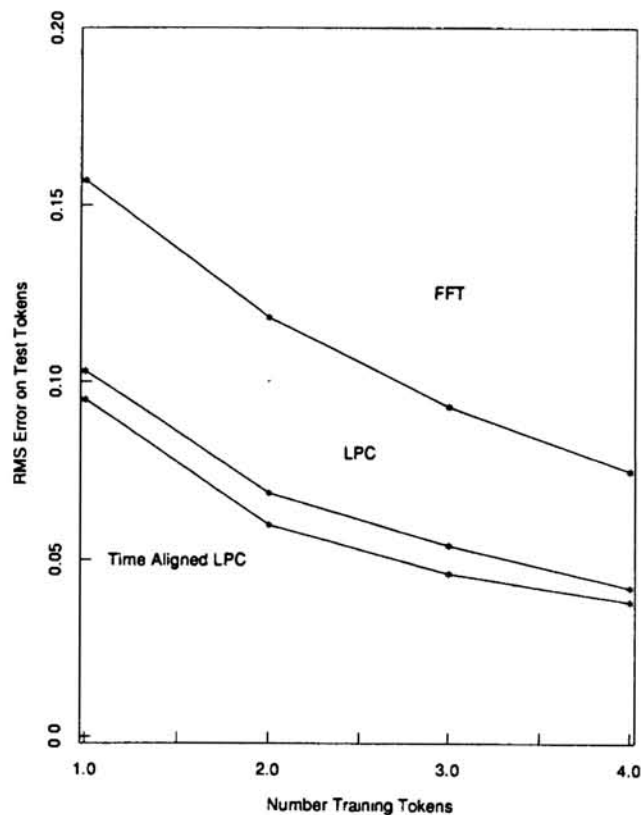

Figure 5. RMS error versus number of training tokens for various input representations.

## E-SET VOCABULARY

The E-Set vocabulary consists of the nine E-words of the English alphabet — B, C, D, E, G, P, T, V, Z. Twenty tokens of each of the nine classes were recorded by a single male speaker. To maximize the sizes of training and test sets, half were used for training and the other half for testing. Ten permutations produced a total of 900 separate recognition trials.

Figure 6 shows typical LPC templates for the nine classes. Notice the double formant ridge due to the "E" sound, which is common to all tokens. Another characteristic feature is the F0 ridge — the upward fold on the left of all tokens which characterizes voicing or pitched sound.

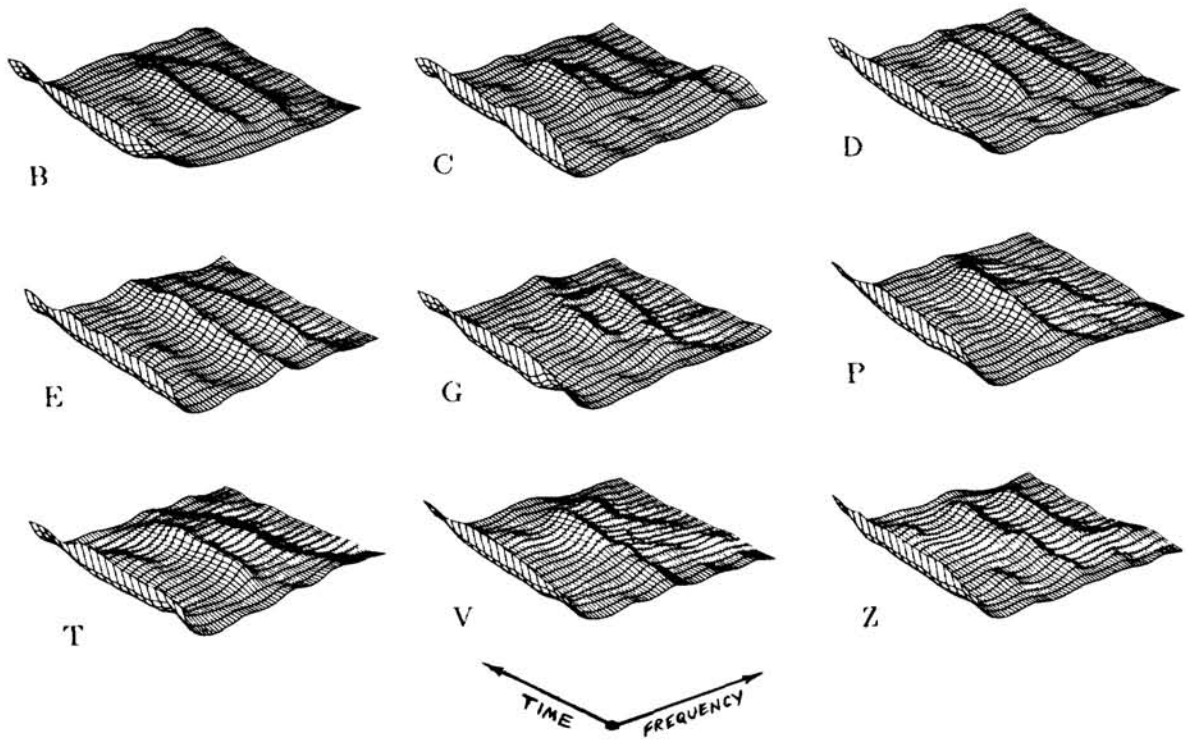

Figure 6. LPC time-frequency plots for representative tokens of the E-set words.

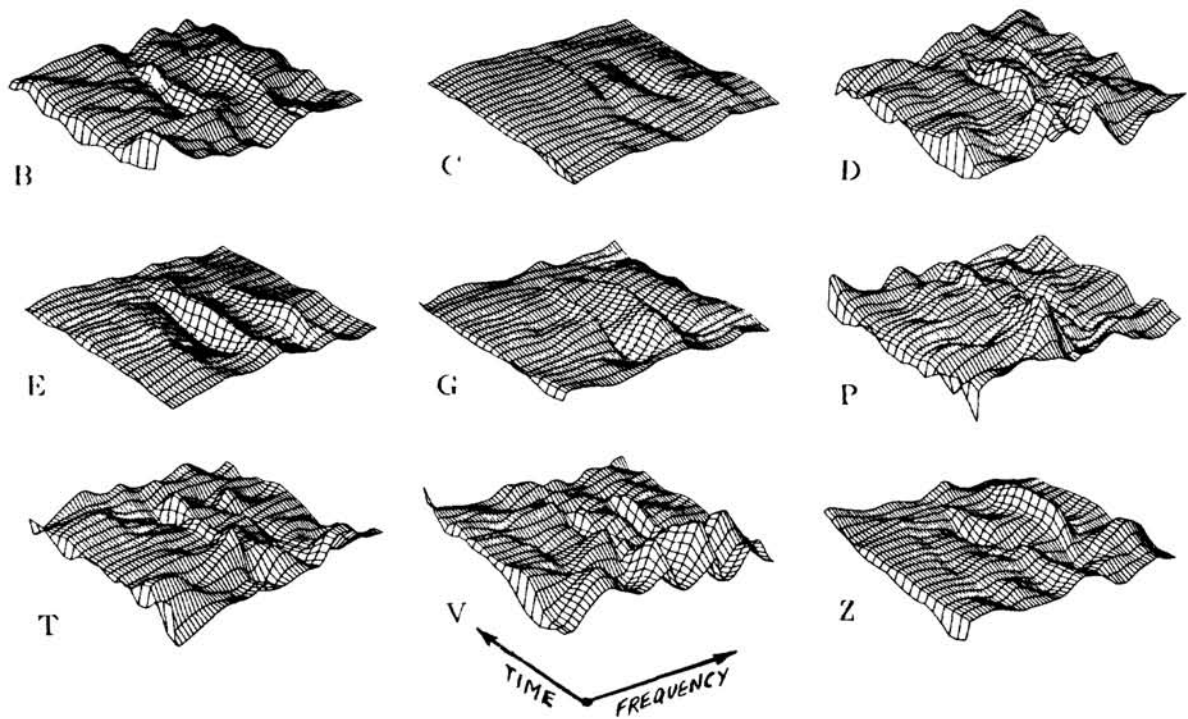

Figure 7. Time-frequency plots of weight values connected to each output neuron "B" through "Z" in a trained perceptron.

Figure 7 shows similar plots illustrating the weights learned by the network when trained on ten tokens of each class. These are plotted like spectra, since one weight is associated with each spectral sample. Note that the patterns have some formant structure. A recognition accuracy of 91.4% included perfect scores for classes C, E, and G.

Notice that weights along the F0 contour are mostly small and some are slightly negative. This is a response to the voiced "E" sound common to all classes. The network has learned to discount "voicing" as a discriminator for this vocabulary.

Notice also the strong "hilly" terrain near the beginning of most templates. This shows where the network has decided to focus much of its discriminating power. Note in particular the hill-valley pair at the beginning of "P" and "T". These are near to formants F2/F3 and could conceivably be formant onset detectors. Note the complicated detector pattern for the "V" sound.

The classes that are easy to discriminate (C, E, G) produce relatively flat and uninteresting weight spaces. A highly convoluted weight space must therefore be correlated with difficulty in discrimination. It makes little sense however that the network should be working hard in the late time ("E" sound) portion of the utterance. Perhaps additional training might reduce this activity, since the network would eventually find little consistent difference there.

A second experiment was conducted to help the network to focus attention. The first k frames of each input token were averaged to produce an average spectrum. These average spectra were then used in a simple nearest neighbor recognizer scheme. Recognition accuracy was measured as a function of k. The highest performance was for k=8, indicating that the first 40% of the word contained most of the "action".

|   | B | C | D | E | G | P | T | V | Z |
|---|---|---|---|---|---|---|---|---|---|
| B | 98 | 0 | 1 | 0 | 0 | 0 | 0 | 1 | 0 |
| C | 0 | 100 | 0 | 0 | 0 | 0 | 0 | 0 | 0 |
| D | 0 | 0 | 98 | 0 | 0 | 2 | 0 | 0 | 0 |
| E | 0 | 0 | 0 | 100 | 0 | 0 | 0 | 0 | 0 |
| G | 0 | 0 | 0 | 0 | 100 | 0 | 0 | 0 | 0 |
| P | 0 | 0 | 3 | 0 | 0 | 93 | 4 | 0 | 0 |
| T | 0 | 0 | 0 | 0 | 0 | 0 | 100 | 0 | 0 |
| V | 2 | 0 | 0 | 0 | 0 | 2 | 0 | 96 | 0 |
| Z | 0 | 0 | 0 | 0 | 0 | 0 | 0 | 1 | 99 |

Figure 8. Confusion matrix of the E-set focused on the first 40% of each word.

All words were resampled to concentrate 20 time frames into the first 40% of the word. LPC spectra were recomputed using a 16th order model and the network was trained on the new templates. Performance increased from 91.4% to 98.2%. There were only 16 classification errors out of the 900 recognition tests. The confusion matrix is shown in Figure 8. Learning times for all experiments consisted of about ten passes through the training set. When weights were primed with average spectral values rather than random values, learning time decreased slightly.

## CONCLUSIONS

Artificial neural networks are capable of high performance in pattern recognition applications, matching or exceeding that of conventional classifiers. We have shown that for difficult speech problems such as time alignment and weak discriminability, artificial neural networks perform at high accuracy exceeding 98%. One-layer perceptrons learn these difficult tasks almost effortlessly — not in spite of their simplicity, but because of it.

## REFERENCES

1. D. J. Burr, "A Neural Network Digit Recognizer", Proceedings of IEEE Conference on Systems, Man, and Cybernetics, Atlanta, GA, October, 1986, pp. 1621-1625.

2. D. J. Burr, "Experiments with a Connectionist Text Reader," IEEE International Conference on Neural Networks, San Diego, CA, June, 1987.

3. M. I. Jordan, "Serial Order: A Parallel Distributed Processing Approach," ICS Report 8604, UCSD Institute for Cognitive Science, La Jolla, CA, May 1986.

4. S. J. Hanson, and D. J. Burr, "What Connectionist Models Learn: Toward a Theory of Representation in Multi-Layered Neural Networks," submitted for publication.

5. W. Y. Huang and R. P. Lippmann, "Comparisons Between Neural Net and Conventional Classifiers," IEEE International Conference on Neural Networks, San Diego, CA, June 21-23, 1987.

6. J. D. Markel and A. H. Gray, Jr., Linear Prediction of Speech, Springer-Verlag, New York, 1976.

7. M. L. Minsky and S. Papert, Perceptrons, MIT Press, Cambridge, Mass., 1969.

8. D. E. Rumelhart, G. E. Hinton, and R. J. Williams, "Learning Internal Representations by Error Propagation," in Parallel Distributed Processing, Vol. 1, D. E. Rumelhart and J. L. McClelland, eds., MIT Press, 1986, pp. 318-362.

9. L. R. Rabiner and M. R. Sambur, "An Algorithm for Determining the Endpoints of Isolated Utterances," BSTJ, Vol. 54, 297-315, Feb. 1975.

10. H. Sakoe and S. Chiba, "Dynamic Programming Optimization for Spoken Word Recognition," IEEE Trans. Acoust., Speech, Signal Processing, Vol. ASSP-26, No. 1, 43-49, Feb. 1978.

11. T. J. Sejnowski and C. R. Rosenberg, "NETtalk: A Parallel Network that Learns to Read Aloud," Technical Report JHU/EECS-86/01, Johns Hopkins University Electrical Engineering and Computer Science, 1986.

12. A. Wieland and R. Leighton, "Geometric Analysis of Neural Network Capabilities," IEEE International Conference on Neural Networks, San Deigo, CA, June 21-24, 1987.
